# Statistics of Natural Images: Scaling in the Woods

**Daniel L. Ruderman\* and William Bialek**
NEC Research Institute
4 Independence Way
Princeton, N.J. 08540

## Abstract

In order to best understand a visual system one should attempt to characterize the natural images it processes. We gather images from the woods and find that these scenes possess an ensemble scale invariance. Further, they are highly non-Gaussian, and this non-Gaussian character cannot be removed through local linear filtering. We find that including a simple "gain control" nonlinearity in the filtering process makes the filter output quite Gaussian, meaning information is maximized at fixed channel variance. Finally, we use the measured power spectrum to place an upper bound on the information conveyed about natural scenes by an array of receptors.

## 1 Introduction

Natural stimuli are playing an increasingly important role in our understanding of sensory processing. This is because a sensory system's ability to perform a task is a statistical quantity which depends on the signal and noise characteristics. Recently several approaches have explored visual processing as it relates to natural images (Atick & Redlich '90, Bialek *et al* '91, van Hateren '92, Laughlin '81, Srinivasan *et al* '82). However, a good characterization of natural scenes is sorely lacking. In this paper we analyze images from the woods in an effort to close this gap. We

further attempt to understand how a biological visual system should best encode these images.

## 2    The Images

Our images consist of $256 \times 256$ pixels $I(\mathbf{x})$ which are calibrated against luminance (see Appendix). We define the image contrast logarithmically as

$$\phi(\mathbf{x}) = \ln(I(\mathbf{x})/I_0),$$

where $I_0$ is a reference intensity defined for each image. We choose this constant such that $\sum_{\mathbf{x}} \phi(\mathbf{x}) = 0$; that is, the average contrast for each image is zero. Our analysis is of the contrast data $\phi(\mathbf{x})$.

## 3    Scaling

Recent measurements (Field '87, Burton & Moorhead '87) suggest that ensembles of natural scenes are scale-invariant. This means that and any quantity defined on a given scale has statistics which are invariant to any change in that scale. This seems sensible in light of the fact that the images are composed of objects at all distances, and so no particular angular scale should stand out. (Note that this does not imply that any particular image is fractal! Rather, the *ensemble* of scenes has statistics which are invariant to scale.)

### 3.1    Distribution of Contrasts

We can test this scaling hypothesis directly by seeing how the statistics of various quantities change with scale. We define the contrast averaged over a box of size $N \times N$ (pixels) to be

$$\phi_N = \frac{1}{N^2} \sum_{i,j=1}^{N} \phi(i,j).$$

We now ask: "How does the probability $P(\phi_N)$ change with $N$?"

In the left graph of figure 1 we plot $\log(P(\phi_N/\phi_N^{RMS}))$ for $N = 1, 2, 4, 8, 16, 32$ along with the parabola corresponding to a Gaussian of the same variance. By dividing out the RMS value we simply plot all the graphs on the same contrast scale. The graphs all lie atop one another, which means the contrast scales—the distribution's shape is invariant to a change in angular scale. Note that the probability is far from Gaussian, as the graphs have linear, and not parabolic, tails. Even after averaging nearly 1000 pixels (in the case of $32 \times 32$), it remains non-Gaussian. This breakdown of the central limit theorem implies that the pixels are correlated over very long distances. This is analogous to the physics of a thermodynamic system at a critical point.

### 3.2    Distribution of Gradients

As another example of scaling, we consider the probability distribution of image gradients. We define the magnitude of the gradient by a discrete approximation

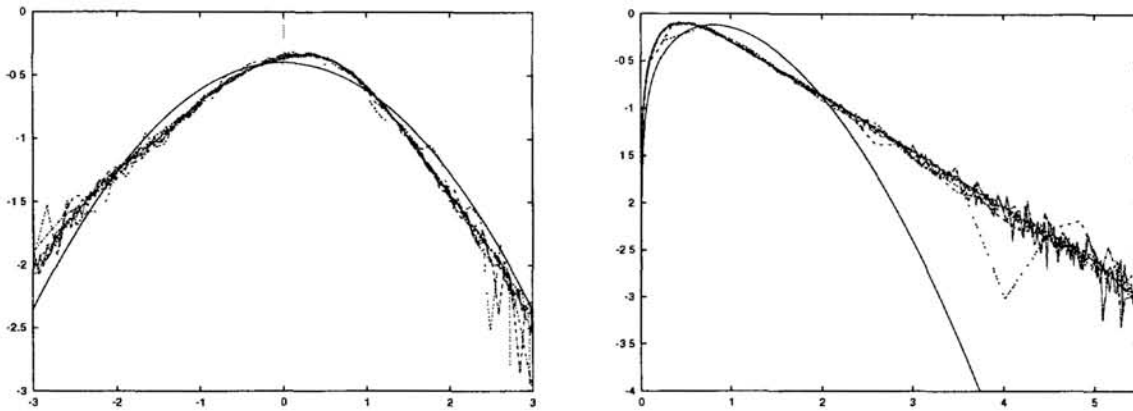

Figure 1: Left: Semi-log plot of $P(\phi_N/\phi_N^{RMS})$ for $N = 1, 2, 4, 8, 16, 32$ with a Gaussian of the same variance for comparison (solid line). Right: Semi-log plot of $P(G_N/\bar{G}_N)$ for same set of $N$'s with a Rayleigh distribution for comparison (solid line).

such that

$$G(\mathbf{x}) = |\mathbf{G}(\mathbf{x})| \approx |\nabla\phi(\mathbf{x})|.$$

We examine this quantity over different scales by first rescaling the images as above and then evaluating the gradient at the new scale. We plot $\log(P(G_N/\bar{G}_N))$ for $N = 1, 2, 4, 8, 16, 32$ in the right graph of figure 1, along with the Rayleigh distribution, $P \approx G\exp(-\alpha G^2)$. If the images had Gaussian statistics, local gradients would be Rayleigh distributed. Note once again scaling of the distribution.

### 3.3  Power Spectrum

Scaling can also be demonstrated at the level of the power spectrum. If the ensemble is scale-invariant, then the spectrum should be of the form

$$S(k) = \frac{A}{k^{2-\eta}},$$

where $k$ is measured in cycles/degree, and $S$ is the power spectrum averaged over orientations.

The spectrum is shown in figure 2 on log-log axes. It displays overlapping data from the two focal lengths, and shows that the spectrum scales over about 2.5 decades in spatial frequency. We determine the parameters as $A = (6.47\pm0.13)\times10^{-3}\text{deg.}^{(0.19)}$ and $\eta = 0.19 \pm 0.01$. The integrated power spectrum up to 60 cycles/degree (the human resolution limit) gives an RMS contrast of about 30%.

## 4  Local Filtering

The early stages of vision consist of neurons which respond to local patches of images. What do the statistics of these local processing units look like? We convolve images with the filter shown in the left of figure 3, and plot the histogram of its output on a semi-log scale on the right of the figure.

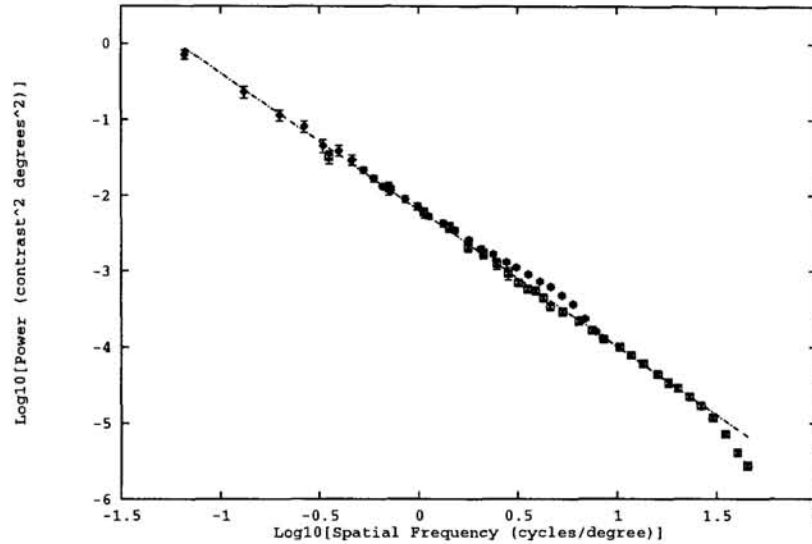

Figure 2: Power spectrum of the contrast of natural scenes (log-log plot).

The distribution is quite exponential over nearly 4 decades in probability. In fact, almost any local linear filter which passes no DC has this property, including center-surround receptive fields. Information theory tells us that it is best to send signals with Gaussian statistics down channels which have power constraints. It is of interest, then, to find some type of filtering which transforms the exponential distributions we find into Gaussian quantities.

Music, as it turns out, has some similar properties. An amplitude histogram from 5 minutes of "The Blue Danube" is shown on the left of figure 4. It is almost precisely exponential over 4 decades in probability. We can guess what causes the excesses over a Gaussian distribution at the peak and the tails; it's the dynamics. When a quiet passage is played the amplitudes lie only near zero, and create the excess in the peak. When the music is loud the fluctuations are large, thus creating the

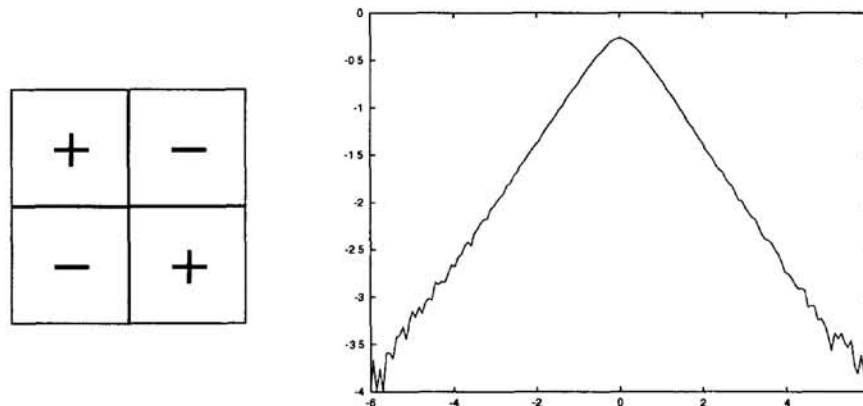

Figure 3: Left: $2 \times 2$ local filter. Right: Semi-log plot of histogram of its output when filtering natural scenes.

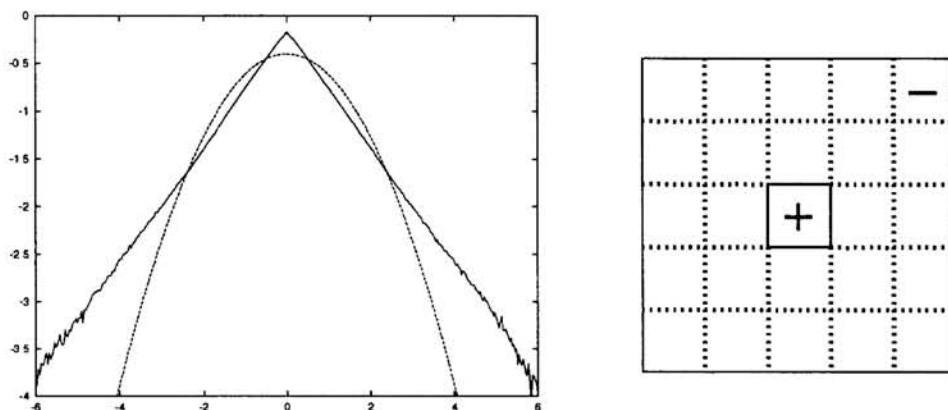

Figure 4: Left: Semi-log histogram of "The Blue Danube" with a Gaussian for comparison (dashed). Right: $5 \times 5$ center-surround filter region.

tails. Most importantly, these quiet and loud passages extend coherently in time; so to remove the peak and tails, we can simply slowly adjust a "volume knob" to normalize the fluctuations. The images are made of objects which have coherent structure over space, and a similar localized dynamic occurs. To remove it, we need some sort of gain control.

To do this, we pass the images through a local filter and then normalize by the local standard deviation of the image (analogous to the volume of a sound passage):

$$\psi(\mathbf{x}) = \frac{\phi(\mathbf{x}) - \bar{\phi}(\mathbf{x})}{\sigma(\mathbf{x})}.$$

Here $\bar{\phi}(\mathbf{x})$ is the mean image contrast in the $N \times N$ region surrounding $\mathbf{x}$, and $\sigma(\mathbf{x})$ is the standard deviation within the same region (see the right of figure 4).

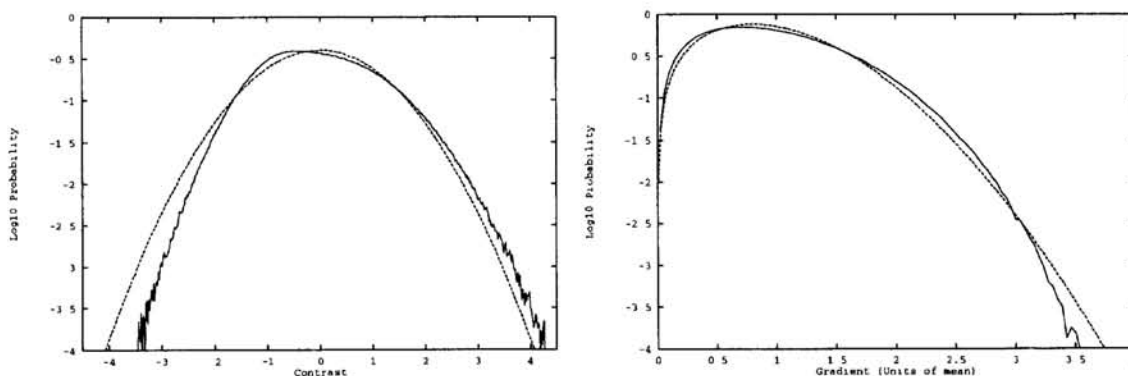

Figure 5: Left: Semi-log plot of histogram of $\psi$, with Gaussian for comparison (dashed). Right: Semi-log plot of histogram of gradients of $\psi$, with Rayleigh distribution shown for comparison (dashed).

We find that for a value $N = 5$ (ratio of the negative surround to the positive center), the histograms of $\psi$ are the closest to Gaussian (see the left of figure 5). Further, the histogram of gradients of $\psi$ is very nearly Rayleigh (see the right of

figure 5). These are both signatures of a Gaussian distribution. Functionally, this "variance normalization" procedure is similar to contrast gain control found in the retina and LGN (Benardete *et al*, '92). Could its role be in "Gaussianizing" the image statistics?

## 5  Information in the Retina

From the measured statistics we can place an upper bound on the amount of information an array of photoreceptors conveys about natural images. We make the following assumptions:

- Images are Gaussian with the measured power spectrum. This places an upper bound on the entropy of natural scenes, and thus an upper bound on the information represented.
- The receptors sample images in a hexagonal array with diffraction-limited optics. There is no aliasing.
- Noise is additive, Gaussian, white, and independent of the image.

The output of the $n^{\text{th}}$ receptor is thus given by

$$y_n = \int d^2x \, \phi(\mathbf{x}) \, M(\mathbf{x} - \mathbf{x}_n) + \eta_n,$$

where $\mathbf{x}_n$ is the location of the receptor, $M(\mathbf{x})$ is the point-spread function of the optics, and $\eta_n$ is the noise. For diffraction-limited optics,

$$M(\mathbf{k}) \approx 1 - |\mathbf{k}|/k_c,$$

where $k_c$ is the cutoff frequency of 60 cycles/degree.

In the limit of an infinite lattice, Fourier components are independent, and the total information is the sum of the information in each component:

$$\mathcal{I} = \frac{A_c}{4\pi} \int_0^{k_c} dk \, k \log\left[1 + \frac{1}{A_c \sigma^2} |M(k)|^2 S(k)\right].$$

Here $\mathcal{I}$ is the information per receptor, $A_c$ is the area of the unit cell in the lattice, and $\sigma^2$ is the variance of the noise.

We take $S(k) = A/k^{2-\eta}$, with $A$ and $\eta$ taking their measured values, and express the noise level in terms of the signal-to-noise ratio in the receptor. In figure 6 we plot the information per receptor as a function of $SNR$ along with the information capacity (per receptor) of the photoreceptor lattice at that $SNR$, which is

$$\mathcal{C} = \frac{1}{2} \log\left[1 + SNR\right].$$

The information conveyed is less than 2 bits per receptor per image, even at $SNR = 1000$. The redundancy of this representation is quite high, as seen by the gap between the curves; at least as much of the information capacity is being wasted as is being used.

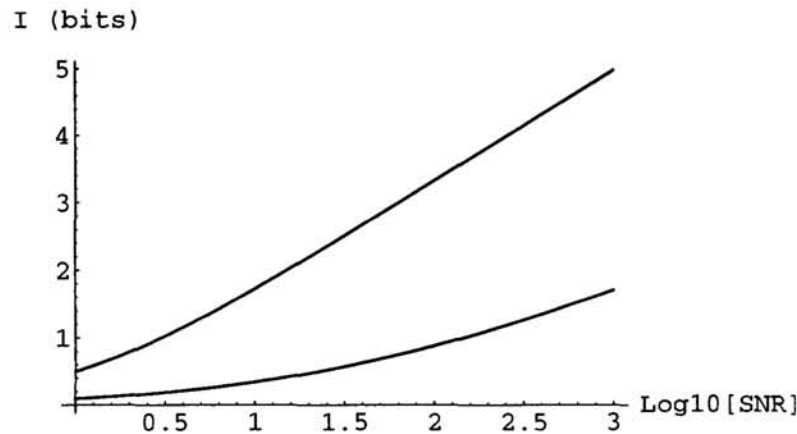

Figure 6: Information per receptor per image (in bits) as a function of $\log(SNR)$ (lower line). Information capacity per receptor (upper line).

## 6   Conclusions

We have shown that images from the forest have scale-invariant, highly non-Gaussian statistics. This is evidenced by the scaling of the non-Gaussian histograms and the power-law form of the power spectrum. Local linear filtering produces values with quite exponential probability distributions. In order to "Gaussianize," we must use a nonlinear filter which acts as a gain control. This is analogous to contrast gain control, which is seen in the mammalian retina. Finally, an array of receptors which encodes these natural images only conveys at most a few bits per receptor per image of information, even at high $SNR$. At an image rate of 50 per second, this places an information requirement of less than about 100 bits per second on a foveal ganglion cell.

## Appendix

Snapshots were gathered using a Sony Mavica MVC-5500 still video camera equipped with a 9.5-123.5mm zoom lens. The red, green, and blue signals were combined according to the standard CIE formula $Y = 0.59\,G + 0.30\,R + 0.11\,B$ to produce a grayscale value at each pixel. The quantity $Y$ was calibrated against incident luminance to produce the image intensity $I(\mathbf{x})$. The images were cropped to the central $256 \times 256$ region.

The dataset consists of 45 images taken at a 15mm focal length (images subtend $15°$ of visual angle) and 25 images at an 80mm focal length ($3°$ of visual angle). All images were of distant objects to avoid problems of focus. Images were chosen by placing the camera at a random point along a path and rotating the field of view until no nearby objects appeared in the frame. The camera was tilted by less than $10°$ up or down in an effort to avoid sky and ground. The forested environment (woods in New Jersey in springtime) consisted mainly of trees, rocks, hillside, and a stream.

## Acknowledgements

We thank H. B. Barlow, B. Gianulis, A. J. Libchaber, M. Potters, R. R. de Ruyter van Stevenink, and A. Schweitzer. Work was supported in part by a fellowship from the Fannie and John Hertz Foundation (to D.L.R.).

## Footnotes

\*Current address: The Physiological Laboratory, Downing Street, Cambridge CB2 3EG, England.

## References

J.J. Atick and N. Redlich. Towards a theory of early visual processing *Neural Computation*, 2:308, 1990.

E. A. Benardete, E. Kaplan, and B. W. Knight. Contrast gain control in the primate retina: P cells are not X-like, some M-cells are. *Vis. Neuosci.*, 8:483-486, 1992.

W. Bialek, D. L. Ruderman, and A. Zee. The optimal sampling of natural images: a design principle for the visual system?, in *Advances in Neural Information Processing systems, 3*, R. P. Lippman, J. E. Moody and D. S. Touretzky, eds., 1991.

G. J. Burton and I. R. Moorhead. Color and spatial structure in natural scenes. *Applied Optics*, 26:157-170, 1987.

D. J. Field. Relations between the statistics of natural images and the response properties of cortical cells. *J. Opt. Soc. Am. A*, 4:2379, 1987.

J. H. van Hateren. Theoretical predictions of spatiotemporal receptive fields of fly LMCs, and experimental validation. *J. Comp. Physiol. A*, 171:157-170, 1992.

S. B. Laughlin. A simple coding procedure enhances a neuron's information capacity. *Z. Naturforsh.*, 36c:910-912, 1981.

M. V. Srinivasan, S. B. Laughlin, and A. Dubs. Predictive coding: a fresh view of inhibition in the retina. *Proc. R. Soc. Lond. B*, 216:427-459, 1982.